# Analyzing human feature learning as nonparametric Bayesian inference

**Joseph L. Austerweil**
Department of Psychology
University of California, Berkeley
Berkeley, CA 94720
Joseph.Austerweil@gmail.com

**Thomas L. Griffiths**
Department of Psychology
University of California, Berkeley
Berkeley, CA 94720
Tom_Griffiths@berkeley.edu

## Abstract

Almost all successful machine learning algorithms and cognitive models require powerful representations capturing the features that are relevant to a particular problem. We draw on recent work in nonparametric Bayesian statistics to define a rational model of human feature learning that forms a featural representation from raw sensory data without pre-specifying the number of features. By comparing how the human perceptual system and our rational model use distributional and category information to infer feature representations, we seek to identify some of the forces that govern the process by which people separate and combine sensory primitives to form features.

## 1 Introduction

Most accounts of the processes underlying human learning, decision-making, and perception assume that stimuli have fixed sets of features. For example, traditional accounts of category learning start with a set of features (e.g., is furry and barks), which are used to learn categories (e.g., dogs). In a sense, features are the basic atoms for these processes. Although the model's features may be combined in particular ways to create new features, the basic primitives are assumed to be fixed. While this assumption has been useful in investigating many cognitive functions, it has been attacked on empirical [1] and theoretical [2] grounds. Experts identify parts of objects in their domain of expertise vastly differently than novices (e.g., [3]), and evidence for flexible feature sets has been found in many laboratory experiments (see [2] for a review). In this paper, we present an account of how flexible features sets could be induced from raw sensory data without requiring the number of features to be prespecified.

From early work demonstrating XOR is only learnable by a linear classifier with the right representation [4] to the so-called "kernel trick" popular in support vector machines [5], forming an appropriate representation is a fundamental issue for applying machine learning algorithms. We draw on the convergence of interest from cognitive psychologists and machine learning researchers to provide a rational analysis of feature learning in the spirit of [6], defining an "ideal" feature learner using ideas from nonparametric Bayesian statistics. Comparing the features identified by this ideal learner to those learned by people provides a way to understand how distributional and category information contribute to feature learning.

We approach the problem of feature learning as one of inferring hidden structure from observed data – a problem that can be solved by applying Bayesian inference. By using methods from nonparametric Bayesian statistics, we can allow an unbounded amount of structure to be expressed in the observed data. For example, nonparametric Bayesian clustering models allow observations to be assigned to a potentially infinite number of clusters, of which only a finite number are represented at any time. When such a model is presented with a new object that it cannot currently explain,

it increases the complexity of its representation to accommodate the object. This flexibility gives nonparametric Bayesian models the potential to explain how people infer rich latent structure from the world, and such models have recently been applied to a variety of aspects of human cognition (e.g., [6, 7]). While nonparametric Bayesian models have traditionally been used to solve problems related to clustering, recent work has resulted in new models that can infer a set of features to represent a set of objects without limiting the number of possible features [8]. These models are based on the Indian Buffet Process (IBP), a stochastic process that can be used to define a prior on the features of objects. We use the IBP as the basis for a rational model of human perceptual feature learning.

The plan of the paper is as follows. Section 2 summarizes previous empirical findings from the human perceptual feature learning literature. Motivated by these results, Section 3 presents a rational analysis of feature learning, focusing on the IBP as one component of a nonparametric Bayesian solution to the problem of finding an optimal representation for some set of observed objects. Section 4 compares human learning and the predictions of the rational model. Section 5 concludes the paper.

## 2   Human perceptual feature learning

One main line of investigation of human feature learning concerns the perceptual learning phenomena of unitization and differentiation. *Unitization* occurs when two or more features that were previously perceived as distinct features merge into one feature. In a visual search experiment by Shiffrin and Lightfoot [9], after learning that the features that generated the observed objects co-vary in particular ways, partcipants represented each object as its own feature instead of as three separate features. In contrast, *differentiation* is when a fused feature splits into new features. For example, color novices cannot distinguish between a color's saturation and brightness; however, people can be trained to make these distinctions [10]. Although general conditions for when differentiation or unitization occur have been outlined, there is no formal account for why and when these processes take place.

In Shiffrin and Lightfoot's visual search experiment [9], participants were trained to find one of the objects shown in Figure 1(a) in a scene where the other three objects were present as distractors. Each object is composed of three features (single line segments) inside a rectangle. The objects can thus be represented by the feature ownership matrix shown in Figure 1(a), with $Z_{ik} = 1$ if object $i$ has feature $k$. After prolonged practice, human performance drastically and suddenly improved, and this advantage did not transfer to other objects created from the same feature set. They concluded that the human perceptual system had come to represent each object holistically, rather than as being composed of its more primitive features. In this case, the fact that the features tended to co-occur only in the configurations corresponding to the four objects provides a strong cue that they may not be the best way to represent these stimuli.

The distribution of potential features over objects provides one cue for inferring a feature representation; however, there can be cases where multiple feature representations are equally good. For example, Pevtzow and Goldstone [11] demonstrated that human perceptual feature learning is affected by category information. In the first part of their experiment, they trained participants to categorize eight "distorted" objects into one of three groups using one of two categorization schemes. The objects were distorted by the addition of a random line segment. The category membership of four of the objects, A-D, depended on the training condition, as shown in Figure 1 (b). Participants in the horizontal categorization condition had objects A and B categorized into one group and objects C and D into the other. Those in the vertical categorization condition learned objects A and C are categorized into one group and objects B and D in the other. The nature of this categorization affected the features learned by participants, providing a basis for selecting one of the two featural representations for these stimuli that would otherwise be equally well-justified based on distributional information.

Recent work has supplemented these empirical results with computational models of human feature learning. One such model is a neural network that incorporates categorization information as it learns to segment objects [2]. Although the inputs to the model are the raw pixel values of the stimuli, the number of features must be specified in advance. This is a serious issue for an analysis of human feature learning because it does not allow us to directly compare different feature set sizes – a critical factor in capturing unitization and differentiation phenomena. Other work has investigated how the human perceptual system learns to group objects that seem to arise from a common cause

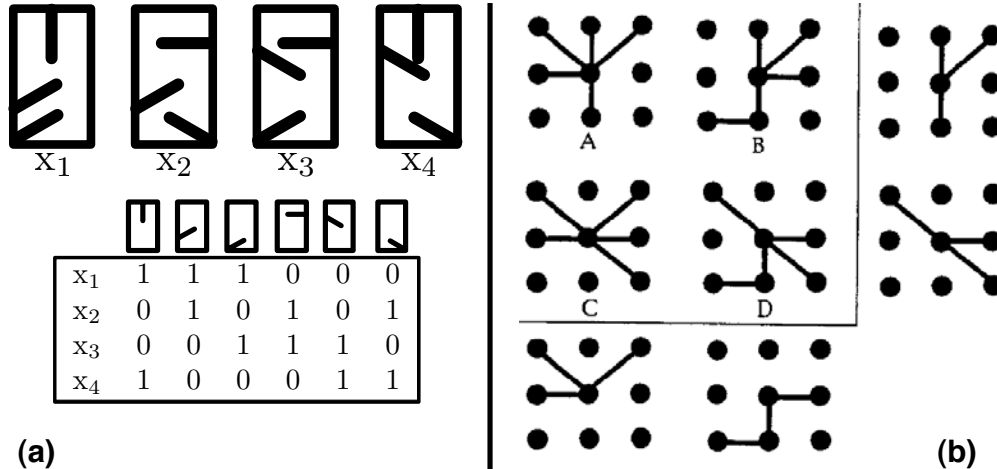

Figure 1: Inferring representations for objects. (a) Stimuli and feature ownership matrix from Shiffrin and Lightfoot [9]. (b) Four objects (A-D) and inferred features depending on categorization scheme from Pevtzow and Goldstone [11]

[12]. This work uses a Bayesian model that can vary the number of causes it identifies, but assumes indifference to the spatial position of the objects and that the basic objects themselves are already known, with a binary variable representing the presence of an object in each scene being given to the model as the observed data. This model is thus given the basic primitives from raw sensory data and does not provide an account of how the human perceptual system identifies these primitives. In the remainder of the paper, we develop a rational model of human feature learning that applies to raw sensory data and does not assume a fixed number of features in advance.

## 3 A Rational Analysis of Feature Learning

Rational analysis is a technique for understanding a cognitive process by comparing it to the optimal solution to an underlying computational problem [6], with the goal of understanding how the structure of this problem influences human behavior. By formally analyzing the problem of inferring featural representations from raw sensory data of objects, we can determine how distributional and category information should influence the features used to represent a set of objects.

### 3.1 Inferring Features from Percepts

Our goal is to form the most probable feature representation for a set of objects given the set of objects we see. Formally, we can represent the features of a set of objects with a feature ownership matrix $Z$ like that shown in Figure 1, where rows correspond to objects, columns correspond to features, and $Z_{ik} = 1$ indicates that object $i$ possesses feature $k$. We can then seek to identify the most likely feature ownership matrix $Z$ given the observed properties of a set of objects $X$ by a simple application of Bayes theorem:

$$\hat{Z} = \arg\max_Z P(Z|X) = \arg\max_Z \frac{P(X|Z)P(Z)}{\sum_{Z'} P(X|Z')P(Z')} = \arg\max_Z P(X|Z)P(Z) \quad (1)$$

This separates the problem of finding the best featural representation given a set of data into two subproblems: finding a representation that is in general probable, as expressed by the prior $P(Z)$, and finding a representation that generates the observed properties of the objects with high probability, as captured by the likelihood $P(X|Z)$. We consider how these distributions are defined in turn.

## 3.2 A Prior on Feature Ownership Matrices

Although in principle any distribution on binary matrices $P(Z)$ could be used as a prior, we use one particular nonparametric Bayesian prior, the Indian Buffet Process (IBP) [8]. The IBP has several nice properties: it allows for multiple features per object, possessing one feature does not make possessing another feature less likely, and it generates binary matrices of unbounded dimensionality. This allows the IBP to use an appropriate, possibly different, number of features for each object and makes it possible for the size of the feature set to be learned from the objects.

The IBP defines a distribution over binary matrices with a fixed number of rows and an infinite number of columns, of which only a finite number are expected to have non-zero elements. The distribution thus permits tractable inference of feature ownership matrices without specifying the number of features ahead of time. The probability of a feature ownership matrix under the IBP is typically described via an elaborate metaphor in which objects are customers and features are dishes in an Indian buffet, with the choice of dishes determining the features of the object, but reduces to

$$P(Z) = \frac{\alpha^{K_+}}{\prod_{h=1}^{2^N-1} K_h!} \exp\{-\alpha H_N\} \prod_{k=1}^{K_+} \frac{(N-m_k)!(m_k-1)!}{N!} \tag{2}$$

where $N$ is the number of objects, $K_h$ is the number of features with history $h$ (the history is the column of the feature interpreted as a binary number), $K_+$ is the number of columns with non-zero entries, $H_N$ is the $N$-th harmonic number, $\alpha$ affects the number of features objects own and $m_k$ is the number of objects that have feature $k$.

## 3.3 Two Likelihood Functions for Perceptual Data

To define the likelihood, we assume $N$ objects with $d$ observed dimensions (e.g., pixels in an image) are grouped in a matrix $X$ ($X = [x_1^T, \ldots, x_N^T]$, where $x_i \in \mathcal{R}^d$). The feature ownership matrix $Z$ marks the commonalities and contrasts between these objects, and the likelihood $P(X|Z)$ expresses how these relationships influence their observed properties. Although in principle many forms are possible for the likelihood, two have been used successfully with the IBP in the past: the linear-Gaussian [8] and noisy-OR [13] models.

The linear-Gaussian model assumes that $x_i$ is drawn from a Gaussian distribution with mean $z_i A$ and covariance matrix $\Sigma_X = \sigma_X^2 I$, where $z_i$ is the binary vector defining the features of object $x_i$ and $A$ is a matrix of the weights of each element of $D$ of the raw data for each feature $k$.

$$p(X|Z, A, \sigma_X) = \frac{1}{(2\pi\sigma_X^2)^{ND/2}} \exp\{-\frac{1}{2\sigma_X^2} \text{tr}((X-ZA)^T(X-ZA))\} \tag{3}$$

Although $A$ actually represents the weights of each feature (which combine with each other to determine raw pixel values of each object), it is integrated out of so that the conditional probability of $X$ given $Z$ and $A$ only depends on $Z$ and hyperparameters corresponding to the variance in $X$ and $A$ (see [8] for details). The result of using this model is a set of images representing the perceptual features corresponding to the matrix $Z$, expressed in terms of the posterior distribution over the weights $A$.

For the noisy-OR model [13], the raw visual data is reduced to binary pixel values. This model assumes that the pixel values $X$ are generated from a noisy-OR distribution where $Z$ defines the features that each object has and $Y$ defines which pixels that should be one for each feature:

$$p(x_{i,d} = 1 | Z, Y, \lambda, \epsilon) = 1 - (1-\lambda)^{z_{i,:} y_{:,d}}(1-\epsilon) \tag{4}$$

where hyperparameters $\epsilon$ and $\lambda$ represent the probability a pixel is turned on without a cause and the probability a feature fails to turn on a pixel respectively. Additionally, $Y$ is assumed to have a Bernoulli prior with hyperparameter $p$ representing the probability that an entry of $Y$ is one, with $p(Y) = \prod_{k,d} p^{y_{k,d}}(1-p)^{1-y_{k,d}}$. The result of using this model is a distribution over binary arrays indicating the pixels associated with the features identified by $Z$, expressed via the posterior distribution on $Y$.

## 3.4 Summary

The prior and likelihood defined in the preceding sections provide the ingredients necessary to use Bayesian inference to identify the features of a set of objects from raw sensory data. The result

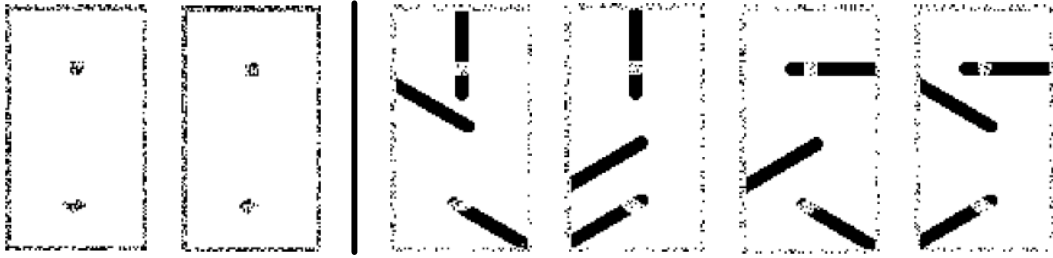

Figure 2: Inferring feature representations using distributional information from Shriffin and Light-foot [9]. On the left, bias features and on the right, the four objects as learned features. The rational model justifies the human perceptual system's unitization of the objects as features

is a posterior distribution on feature ownership matrices $Z$, indicating how a set of objects could be represented, as well as an indication of how the features identified by this representation are expressed in the sensory data. While computing this posterior distribution exactly is intractable, we can use existing algorithms developed for probabilistic inference in these models. Although we used Gibbs sampling – a form of Markov chain Monte Carlo that produces samples from the posterior distribution on $Z$ – for all of our simulations, Reversible Jump MCMC and particle filtering inference algorithms have also been derived for these models [8, 13, 14].

## 4   Comparison with Human Feature Learning

The nonparametric Bayesian model outlined in the previous section provides an answer to the question of how an ideal learner should represent a set of objects in terms of features. In this section we compare the representations discovered by this ideal model to human inferences. First, we demonstrate that the representation discovered by participants in Shiffrin and Lightfoot's experiment [9] is optimal under this model. Second, we illustrate that both the IBP and the human perceptual system incorporate category information appropriately. Finally, we present simulations that show the flexibility of the IBP to learn different featural representations depending on the distributional information of the actual features used to generate the objects, and discuss how this relates to the phenomena of unitization and differentiation more generally.

### 4.1   Using Distributional Information

When should whole objects or line segments be learned as features? It is clear which features should be learned when all of the line segments occur independently and when the line segments in each object always occur together (the line segments and the objects respectively). However, in the inter-mediate cases of non-perfect co-occurence, what should be learned? Without a formal account of feature learning, there is no basis for determining when object "wholes" or "parts" should be learned as features. Our rational model provides an answer – when there is enough statistical evidence for the individual line segments to be features, then each line segment should be differentiated into features. Otherwise, the collection of line segments should be learned as one unitized feature.

The stimuli constructed by Shiffrin and Lightfoot [9] constitute one of the intermediate cases between the extremes of total independence and perfect correlation, and are thus a context in which formal modeling can be informative. Figure 2 presents the features learned by applying the model with a noisy-OR likelihod to this object set. The features on left are the bias and the four features on the right are the four objects from their study. The learned features match the representation formed by people in the experiment. Although there is imperfect co-occurence between the features in each object, there is not enough statistical evidence to warrant representing the object as a combination of features. These results were obtained with an object set consisting of five copies of each of the four objects with added noise that flips a pixel's value with probability $\frac{1}{75}$. The results were obtained by running the Gibbs sampler with initialization $p = 0.2, \alpha = 1.0, \epsilon = 0.025$, and $\lambda = .975$. Inference is robust to different initializations as long as they are near these values.

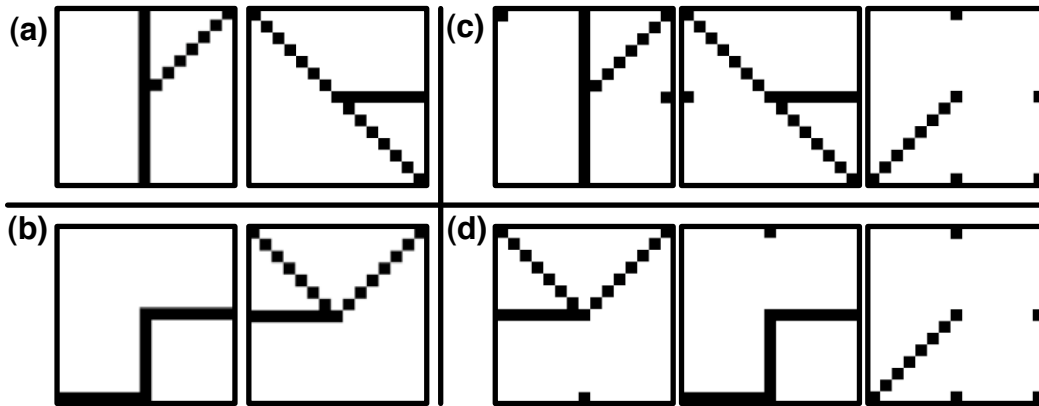

Figure 3: Inferring feature representations using category information from Pevtzow and Goldstone [11]. (a) - (b) Features learned from using the rational model with the noisy-OR likelihood where 10 distorted copies of objects A-D comprise the object set with (a) horizontal and (b) vertical categorization schemes ($c = 35$) respectively. The features inferred by the model match those learned by participants in the experiment. (c) - (d) Features learned from using the same model with the full object set with 10 distorted copies of each object, the (c) horizontal and (d) vertical categorization schemes ($c = 75$) respectively. The first two features learned by the model match those learned by participants in the experiment. The third feature represents the intersection of the third category (Pevtzow and Goldstone did not test if participants learned this feature).

## 4.2   Using Category Information

To model the results of Pevtzow and Goldstone [11], we applied the rational model with the noisy-OR likelihood to the stimuli used in their experiment. Although this model does not incorporate category information directly, we included it indirectly by postpending $c$ bits per category to the end of each image. Figure 3 (a) and (b) show the features learned by the model when trained on distorted objects A-D using both categorization schemes. The categorization information is used appropriately by the model and mirrors the different feature representations inferred by the two pariticipant groups. Figure 3 (c) and (d) show the features learned by the model when given ten distorted copies of all eight objects. Like the human perceptual system, the model infers different, otherwise undistinguishable, feature sets using categorization information appropriately. Although the neural network model of feature learning presented in [2] also inferred correct representations with the four object set, this model did not produce correct results for the eight object set. Inference is susceptible to local minima given poor initializations of the hyperparameters. The features shown in Figure 3 used the following initialization: $p = 0.125, \alpha = 1.5, \lambda = 0.99$, and $\epsilon = 0.01$.[1]

## 4.3   Unitization and Differentiation

The results presented in this section show that our rational model reproduces human inferences for particular datasets, suggesting that the model might be useful more generally in identifying conditions under which the human perceptual system should unitize or differentiate sensory primitives. The Shiffrin and Lightfoot results demonstrated one case where whole objects should be learned as features even though each object was created from features that did not perfectly co-occur. The IBP confirms the intuitive explanation that there is not enough statistical evidence to break (differentiate) the objects into individual features and thus the unitization behavior of the participants is justified. However, there is no comparison with the same underlying feature set to when statistical evidence warrants differentiation, so that the individual features should be learned as features.

To illustrate the importance of distributional information on the inferred featural representation, we designed a simulation to show cases where the objects and the actual features used to generate the objects should be learned as the features. Figure 4 (a) shows the bias (on left) and the set of six features used in the simulations. Figure 4 (b) is an artificially generated set of observed objects for

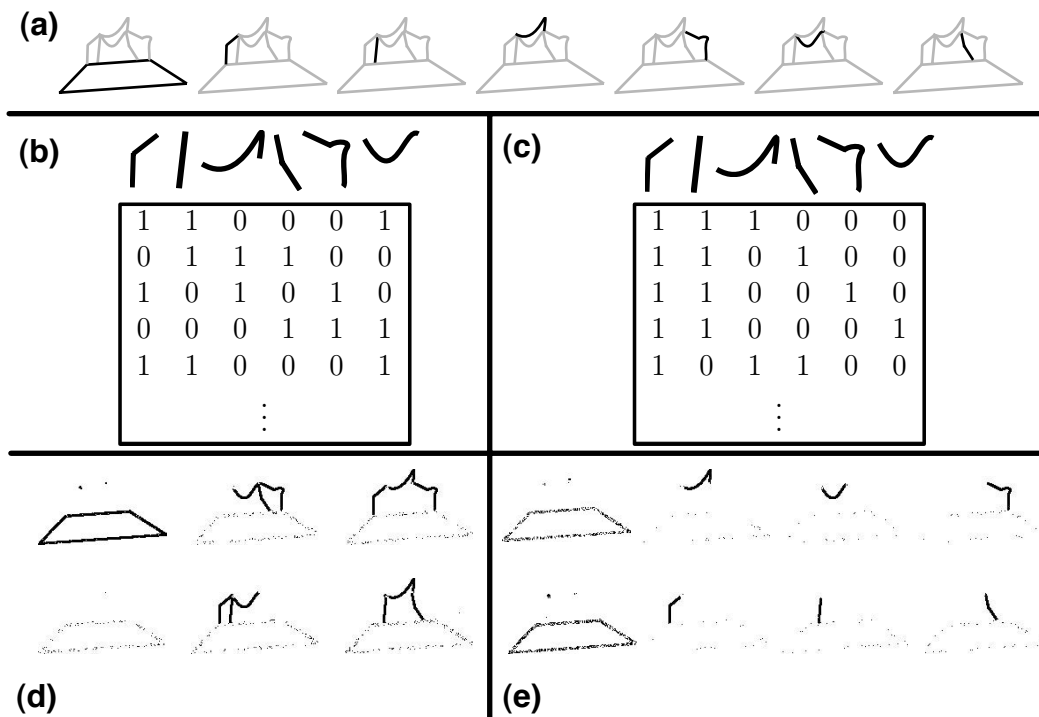

Figure 4: Inferring different feature representations depending on the distributional information. (a) The bias (on left) and the six features used to generate both object sets. (b) - (c) The feature membership matrices for (b) *unitization* and (c) *differentiation* sets respectively. (d) - (e) The feature representations inferred by model for (d) *unitization* and (e) *differentiation* sets respectively.

which there is not enough statistical evidence to warrant differentiation. This is the same underlying feature membership matrix as the Shiffrin and Lightfoot result (*unitization* set). Figure 4 (c) is an artificially generated object set in which the observed objects should be differentiated. Here, the features used to generate the objects occur independently of each other and thus the underlying feature membership matrix used to generate the observed objects is all possible $\binom{6}{3}$ objects (*differentiation* set).

Figure 4 (d) and (e) show the results of applying the rational model with a noisy-OR likelihood to these two object sets. When the underlying features occur independently of each other, the model represents the objects in terms of these features. When the features often co-occur, the model forms a representation which consists simply of the objects themselves. For each simulation, 40 objects from the appropriate set (repeating as necessary) were presented to the model. Each object was perturbed by added noise that flipped a pixel's value with probability $\frac{1}{75}$. The hyperparameters were inferred with Metropolis-Hastings steps during Gibbs sampling and were initialized to: $\alpha = 1$, $\sigma_X^2 = 2.25$, and $\sigma_A^2 = 0.5$. These simulations demonstrate that even when the same underlying features create two object sets, different representations should be inferred depending on the the distributional information, suggesting that this kind of information can be a powerful driving force behind unitization and differentiation.

## 5 Discussion and Future Directions

The flexibility of human featural representations and the power of representation in machine learning make a formal account of how people derive representations from raw sensory information tremendously important. We have outlined one approach to this problem, drawing on ideas from nonparametric Bayesian statistics to provide a rational account of how the human perceptual system uses distributional and category information to infer representations. First, we showed that in one circumstance where it is ambiguous whether or not parts or objects should form the featural rep-

resentation of the objects, that this model peforms similarly to the human perceptual system (they both learn the objects themselves as the basic features). Second, we demonstrated that the IBP and the human perceptual systems both use categorization information to make the same inductions as appropriate for the given categorization scheme. Third, we further investigated how distributional information of the features that create the object set affects the inferred representation. These results begin to sketch a picture of human feature learning as a rational combination of different sources of information about the structure of a set of objects.

There are two main future directions for our work. First, we intend to perform further analysis of how the human perceptual system uses statistical cues. Specifically, we plan to investigate whether the feature sets identified by the perceptual system are affected by the distributional information it is given (as our simulations would suggest). Second, we hope to use hierarchical nonparametric Bayesian models to investigate the interplay between knowledge effects and perceptual input. Recent work has identified a connection between the IBP and the Beta process [15], making it possible to define hierarchical Bayesian models in which the IBP appears as a component. Such models would provide a more natural way to capture the influence of category information on feature learning, extending the analyses that we have performed here.

**Acknowledgements** We thank Rob Goldstone, Karen Schloss, Stephen Palmer, and the Computational Cognitive Science Lab at Berkeley for discussions and the Air Force Office of Scientific Research for support.

## Footnotes

[1]The features inferred by the model in each figure has highest probability given the images it observed.

# References

[1] P. G. Schyns, R. L. Goldstone, and J. Thibaut. Development of features in object concepts. *Behavioral and Brain Sciences*, 21:1–54, 1998.

[2] R. L. Goldstone. Learning to perceive while perceiving to learn. In *Perceptual organization in vision: Behavioral and neural perspectives*, pages 233–278. 2003.

[3] I. Biederman and M. M. Schiffrar. Sexing day-old chicks: A case study and expert systems analysis of a difficult perceptual-learning task. *Journal of Experimental Psychology: Learning, Memory, and Cognition*, 13:640–645, 1987.

[4] M. L. Minsky and S. A. Papert. *Perceptrons*. MIT Press, Cambridge, MA, 1969.

[5] B. Scholkopf and A. J. Smola. *Learning with Kernels*. MIT Press, Cambridge, MA, 2001.

[6] J. R. Anderson. Is human cognition adaptive? *Behavioral and Brain Sciences*, 14:471–517, 1991.

[7] A. N. Sanborn, T. L. Griffiths, and D. J. Navarro. A more rational model of categorization. In *Proceedings of the 28th Annual Conference of the Cognitive Science Society*, 2006.

[8] T. L. Griffiths and Z. Ghahramani. Infinite latent feature models and the Indian buffet process. In *Advances in Neural Information Processing Systems 18*, 2006.

[9] R. M. Shiffrin and N. Lightfoot. Perceptual learning of alphanumeric-like characters. In *The psychology of learning and motivation*, volume 36, pages 45–82. Academic Press, San Diego, 1997.

[10] R. L. Goldstone. Influences of categorization on perceptual discrimination. *Journal of Experimental Psychology: General*, 123:178–200, 1994.

[11] R. Pevtzow and R. L. Goldstone. Categorization and the parsing of objects. In *Proceedings of the Sixteenth Annual Conference of the Cognitive Science Society*, pages 712–722, Hillsdale, NJ, 1994. Lawrence Erlbaum Associates.

[12] G. Orban, J. Fiser, R. N. Aslin, and M. Lengyel. Bayesian model learning in human visual perception. In *Advances in Neural Information Processing Systems 18*, 2006.

[13] F. Wood, T. L. Griffiths, and Z. Ghahramani. A non-parametric Bayesian method for inferring hidden causes. In *Proceeding of the 22nd Conference on Uncertainty in Artificial Intelligence*, 2006.

[14] F. Wood and T. L. Griffiths. Particle filtering for nonparametric Bayesian matrix factorization. In *Advances in Neural Information Processing Systems 19*, 2007.

[15] R. Thibaux and M. I. Jordan. Hierarchical Beta processes and the Indian buffet process. Technical Report 719, University of California, Berkeley. Department of Statistics, 2006.

